# Distribution Matching for Transduction

**Novi Quadrianto**
RSISE, ANU & SML, NICTA
Canberra, ACT, Australia
novi.quad@gmail.com

**James Petterson**
RSISE, ANU & SML, NICTA
Canberra, ACT, Australia
james.petterson@nicta.com.au

**Alex J. Smola**
Yahoo! Research
Santa Clara, CA, USA
alex@smola.org

## Abstract

Many transductive inference algorithms assume that distributions over training and test estimates should be related, e.g. by providing a large margin of separation on both sets. We use this idea to design a transduction algorithm which can be used without modification for classification, regression, and structured estimation. At its heart we exploit the fact that for a good learner the distributions over the outputs on training and test sets should match. This is a classical two-sample problem which can be solved efficiently in its most general form by using distance measures in Hilbert Space. It turns out that a number of existing heuristics can be viewed as special cases of our approach.

## 1   Introduction

Transduction relies on the fundamental assumption that training and test data should exhibit similar behavior. For instance, in large margin classification a popular concept is to assume that both training and test data should be separable with a large margin [4]. A similar matching assumption is made by [8, 15] in requiring that class means are balanced between training and test set. Corresponding distributional assumptions are made for classification by [5], for regression by [10], and in the context of sufficient statistics on the marginal polytope by [3, 6].

Such matching assumptions are well founded: after all, we assume that both training data $X = \{x_1, \ldots, x_m\} \subseteq \mathfrak{X}$ and test data $X' := \{x'_1, \ldots, x'_{m'}\} \subseteq \mathfrak{X}$ are drawn independently and identically distributed from the *same* distribution $p(x)$ on a domain $\mathfrak{X}$. It therefore follows that for any function (or set of functions) $f : \mathfrak{X} \to \mathbb{R}$ the distribution of $f(x)$ where $x \sim p(x)$ should also behave in the same way on both training and test set. Note that this is *not* automatically true if we get to choose $f$ *after* seeing $X$ and $X'$.

Rather than *indirectly* incorporating distributional similarity, e.g. by a large margin heuristic, we cast this goal as a *two-sample problem* which will allow us to draw on a rich body of literature for comparing distributions. One advantage of our setting is its full generality. That is, it is applicable without much need for customization to all estimation problems, whether structured or not. Furthermore, our approach is scalable and can be used easily with online optimization algorithms requiring no additional storage and only an additional $O(1)$ computation per observation. This allows us to perform a multi-category classification on a dataset with $3.2 \cdot 10^6$ observations. At its heart it uses the following: rather than minimizing only the empirical risk, regularized risk, log-posterior, or related quantities obtained only on the training set, let us add a divergence term characterizing the mismatch in distributions between training and test set. We show that the Maximum-Mean-Discrepancy [7] is a suitable quantity for this purpose. Moreover, we show that for certain choices of kernels we are able to recover a number of existing transduction constraints as a special case.

Note that our setting is entirely complementary to the notion of modifying the function space due to the availability of additional data. The latter stream of research led to the use of graph kernels and similar density-related algorithms [1]. It is often referred to as the cluster assumption in semi-supervised learning. In other words, both methods can be combined as needed. That said, while

distribution matching *always* holds thus making our method always applicable, it is not entirely clear whether the cluster assumption is always satisfied (e.g. assume a noisy classification problem).

Distribution matching, however, comes with a nontrivial price: the objective of the optimization problem ceases to be convex except for rather special cases (which correspond to algorithms that have been proposed as previous work). While this is a downside, it is a property inherent in most transduction algorithms — after all, we are dealing with algorithms to obtain self-consistent labelings, predictions, or regressions on the data and there may exist more than one potential solution.

## 2 The Model

**Supervised Learning** Denote by $\mathcal{X}$ and $\mathcal{Y}$ the domains of data and labels and let $\Pr(x, y)$ be a distribution on $\mathcal{X} \times \mathcal{Y}$ from which we are drawing observations. Moreover, denote by $X, Y$ sets of data and labels of the training set and by $X', Y'$ test data and labels respectively. In general, when designing an estimator one attempts to minimize some regularized risk functional

$$R_{\text{reg}}[f, X, Y] := \frac{1}{m} \sum_{i=1}^{m} l(x_i, y_i, f) + \lambda \Omega[f] \tag{1}$$

or alternatively (in a Bayesian setting) one deals with a log-posterior probability

$$\log p(f|X, Y) = \sum_{i=1}^{m} \log p(y_i|x_i, f) + \log p(f) + \text{const.} \tag{2}$$

Here $p(f)$ is the prior of the parameter choice $f$ and $p(y_i|x_i, f)$ denotes the likelihood. $f$ typically is a mapping $\mathcal{X} \to \mathbb{R}$ (for scalar problems such as regression or classification) or $\mathcal{X} \to \mathbb{R}^d$ (for multivariate problems such as named entity tagging, image annotation, matching, ranking, or more generally the clique potentials of graphical models). Note that we are free to choose $f$ from one of many function classes such as decision trees, neural networks, or (nonparametric) linear models. The specific choice boils down to the ability to control the complexity of $f$ efficiently, to one's prior knowledge of what constitutes a simple function, to runtime constraints, and to the availability of scalable algorithms. In general, we will denote the training-data dependent term by

$$R_{\text{train}}[f, X, Y] \tag{3}$$

and we assume that finding some $f$ for which $R_{\text{train}}[f, X, Y]$ is small is desirable. An analogous reasoning applies to sampling-based algorithms, however we skip them for the sake of conciseness.

**Distribution Matching** Denote by $f(X) := \{f(x_1), \ldots, f(x_m)\}$ and by $f(X') := \{f(x'_1), \ldots, f(x'_{m'})\}$ the applications of our estimator (and any related quantities) to training and test set respectively. For $f$ chosen a-priori, the distributions from which $f(X)$ and $f(X')$ are drawn coincide. Clearly, this *should* also hold whenever $f$ is chosen by an estimation process. After all, we want that the empirical risk on the training and test sets match. While this cannot be checked directly, we can at least check closeness between the distributions of $f(x)$. This reasoning leads us to the following additional term for the objective function of a transduction problem:

$$D(f(X), f(X')) \tag{4}$$

Here $D(f(X), f(X'))$ denotes the distance between the two distributions $f(X)$ and $f(X')$. This leads to an overall objective for learning

$$R_{\text{train}}[f, X, Y] + \gamma D(f(X), f(X')) \text{ for some } \gamma > 0 \tag{5}$$

when performing transductive inference. For instance, we could use the Kolmogorov-Smirnov statistic between both sets as our criterion, that is, we could use

$$D(f(X), f(X')) = \|F(f(X)) - F(f(X'))\|_{\infty} \tag{6}$$

the $L_{\infty}$ norm between the cumulative distribution functions $F$ associated with the empirical distributions $f(X)$ and $f(X')$ to quantify the differences between both distributions. The problem with the above choice of distance is that it is *not* easily computable: we first need to evaluate $f$ on both $X$ and $X'$, then sort the arguments, and finally compute the largest deviation between both sets before

we can even attempt computing gradients or using a similar optimization procedure. Such a choice is clearly computationally undesirable.

Instead, we propose the following: denote by $\mathcal{H}$ a Reproducing Kernel Hilbert Space with kernel $k$ defined on $\mathcal{X}$. In this case one can show [7] that whenever $k$ is characteristic (or universal), the map

$$\mu : p \to \mu[p] := \mathbf{E}_{x \sim p(x)}[k(x, \cdot)] \text{ with associated distance } D(p, p') := \|\mu[p] - \mu[p']\|^2 \quad (7)$$

characterizes a distribution uniquely. Examples of a characteristic kernel is Gaussian RBF, Laplacian and $B_{2n+1}$-splines. It is possible to design online estimates of the distance quantity which can be used for fast two-sample tests between $\mu[X]$ and $\mu[X']$. Details on how this can be achieved are deferred to Section 4.

## 3 Special Cases

Before discussing a specific algorithm let us consider a number of special cases to show that this basic idea is rather common in the literature (albeit not as explicit as in the present paper).

**Mean Matching for Classification**  Joachims [8] uses the following balancing constraint in the objective function of a binary classifier where $\hat{y}(x) = \mathrm{sgn}(f(x))$ for $f(x) = \langle w, x \rangle$. In order to balance the outputs between training and test set, [8] imposes the linear constraint

$$\frac{1}{m} \sum_{i=1}^{m} f(x_i) = \frac{1}{m'} \sum_{i=1}^{m'} f(x_i'). \quad (8)$$

Assuming a linear kernel $k$ on $\mathbb{R}$ this constraint is equivalent to requiring that

$$\mu[f(X)] = \frac{1}{m} \sum_{i=1}^{m} \langle f(x_i), \cdot \rangle = \frac{1}{m'} \sum_{i=1}^{m'} \langle f(x_i'), \cdot \rangle = \mu[f(X')]. \quad (9)$$

Note that [8] uses the margin distribution as an additional criterion which will be discussed later.

This setting can be extended to multiclass categorization and estimation with structured random variables in a straightforward fashion [15] simply by requiring a constraint corresponding to (9) to be satisfied for all possible values of $y$ via

$$\frac{1}{m} \sum_{i=1}^{m} \langle f(x_i, y), \cdot \rangle = \frac{1}{m'} \sum_{i=1}^{m'} \langle f(x_i', y), \cdot \rangle \text{ for all } y \in \mathcal{Y}. \quad (10)$$

This is equivalent to a linear kernel on $\mathbb{R}^{\mathcal{Y}}$ and the requirement that the distributions of the values $f(x, y)$ match for all $y$.

**Distribution Matching for Classification**  Gärtner et. al. [5] propose to perform transduction by requiring that the conditional class probabilities on training and test set match. That is, for classifiers generating a distribution of the form $y_i' \sim p(y_i'|x_i', w)$ they require that the marginal class probability on the test set matches the empirical class probability on the training set. Again, this can be cast in terms of distribution matching via

$$\mu[g \circ f(X)] = \frac{1}{m} \sum_{i=1}^{m} \langle g \circ f(x_i), \cdot \rangle = \frac{1}{m'} \sum_{i=1}^{m'} \langle g \circ f(x_i'), \cdot \rangle = \mu[g \circ f(X')]$$

Here $g(\chi) = \frac{1}{1+e^{-\chi}}$ denotes the likelihood of $y = 1$ in logistic regression for the model $p(y|\chi) = \frac{1}{1+e^{-y\chi}}$. Note that instead of choosing the logistic transform $g$ we could have picked a large number of other transformations. Indeed, we may strengthen the requirement above to hold for all $g$ in some given function class $\mathcal{G}$ as follows:

$$D(f(X), f(X')) := \sup_{g \in \mathcal{G}} \left[ \frac{1}{m} \sum_{i=1}^{m} g \circ f(x_i) - \frac{1}{m'} \sum_{i=1}^{m'} g \circ f(x_i') \right] \quad (11)$$

If we restrict ourselves to $g$ having bounded norm in a Reproducing Kernel Hilbert Space we obtain exactly the criterion (7). Gretton et. al. [7] show by duality that this is equivalent to the distance proposed in (11). In other words, generalizing distribution matching to apply to transforms other than the logistic leads us directly to our new transduction criterion.

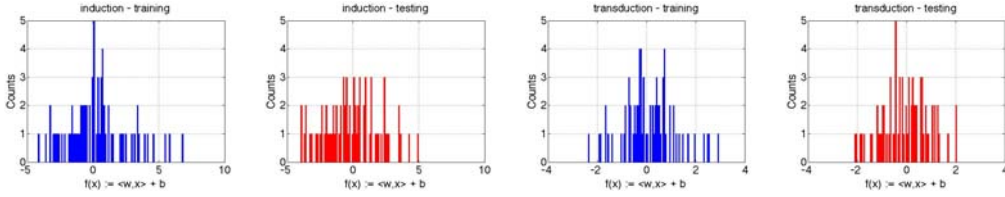

Figure 1: Score distribution of $f(x) = \langle w, x \rangle + b$ on the 'iris' toy dataset. From left to right: induction scores on the training set; test set; transduction scores on the training set; test set; Note that while the margin distributions on training and test set are very different for induction, the ones for transduction match rather well. It results in a 10% reduction of the misclassification error.

**Distribution Matching for Regression**    A similar idea for transduction was proposed by [10] in the context of regression: requiring that both *means* and predictive *variances* of the estimate agree between training and test set. For a heteroscedastic regression estimate this constraint between training and test set is met simply by ensuring that the distributions over first and second order moments of a Gaussian exponential family distribution match. The same goal can be achieved by using a polynomial kernel of second degree on the estimates, which shows that regression transduction can be viewed as a special case.

**Large Margin Hypothesis**    A key assumption in transduction is that a good hypothesis is characterized by a *large margin* of separation on both training and test set. Typically, the latter is enforced by some nonconvex function, e.g. of the form $\max(0, 1 - |f(x)|)$, thus leading to a nonconvex optimization problem. Generalizations of this approach to multiclass and structured estimation settings is not entirely trivial and requires a number of heuristic choices (e.g. how to define the equivalent of the hat function $\max(0, 1 - |\chi|)$ that is commonly used in binary transduction).

Instead, if we require that the distribution of values $f(x, \cdot)$ on $X'$ match those on $X$, we *automatically* obtain a loss function which enforces the large margin hypothesis whenever it is actually *achievable* on the training set. After all, assume that $f(X)$ exhibits a large margin of separation whereas $f(X')$ does not. In this case, $D(f(X), f(X'))$ is large and we obtain better risk minimizers by minimizing the discrepancy of the distributions. The key point is that by using a two-sample criterion it is possible to obtain such criteria *automatically* without the need for heuristic choices. See Figure 1 for illustrations of this idea.

## 4   Algorithm

**Streaming Approximation**    In general, minimizing $D(f(X), f(X'))$ is computationally infeasible since the estimation of the distributional distance requires access to $f(X)$ and $f(X')$ rather than evaluations on a small sample. However, for Hilbert-Space based distance measures it is possible to find an online estimate of $D$ as follows [7]:

$$D(p, p') := \|\mu[p] - \mu[p']\|^2 = \left\| \mathbf{E}_{x \sim p(x)}[k(x, \cdot)] - \mathbf{E}_{x' \sim p'(x')}[k(x', \cdot)] \right\| \tag{12}$$

$$= \mathbf{E}_{x, \tilde{x} \sim p} \mathbf{E}_{x', \tilde{x}' \sim p'}[k(x, \tilde{x}) - k(x, \tilde{x}') - k(\tilde{x}, x') + k(x', \tilde{x}')] \tag{13}$$

The symbol $\tilde{(.)}$ denotes a second set of observations drawn from the same distribution. Note that (13) decomposes into a sum over 4 kernel functions, each of which takes as arguments a pair of instances drawn from $p$ and $p'$ respectively. Hence we can find an unbiased estimate via

$$\hat{D} := \frac{1}{m} \sum_{i=1}^{m} D_i \text{ where}$$

$$D_i := [k(f(x_i), f(x_{i+1})) - k(f(x_i), f(x'_{i+1})) - k(f(x_{i+1}), f(x'_i)) + k(f(x'_i), f(x'_{i+1}))] \tag{14}$$

under the assumption that $X$ and $X'$ contain iid data. Note that the assumption automatically fails if there is sequential dependence within the sets $X$ or $X'$ (e.g. we see all positive labels before we see the negative ones). In this case it is necessary to randomize $X$ and $X'$.

**Stochastic Gradient Descent** The fact that the estimator of the distance $\hat{D}$ decomposes into an average over a function of pairs from the training and test set respectively means that we can use $D_i$ as a stochastic approximation. Applying the same reasoning to the loss function in the regularized risk (1) we obtain the following loss

$$\bar{l}(x_i, x_{i+1}, y_i, y_{i+1}, x'_i, x'_{i+1}, f) \tag{15}$$
$$:= l(x_i, y_i, f) + l(x_{i+1}, y_{i+1}, f) + 2\lambda\Omega[f] +$$
$$\gamma[k(f(x_i), f(x_{i+1})) - k(f(x_i), f(x'_{i+1})) - k(f(x_{i+1}), f(x'_i)) + k(f(x'_i), f(x'_{i+1}))]$$

as a stochastic estimate of the objective function defined in (5). This suggests Algorithm 1, which is a nonconvex variant of [12]. Note that at *no* time we need to store past data even for computing the distance between both distributions.

---

**Algorithm 1** Stochastic Gradient Descent

---

**Input:** Convex set $A$, objective function $\bar{l}$
Initialize $w = 0$
**for** $t = 1$ to $N$ **do**
    Sample $(x_i, y_i), (x_{i+1}, y_{i+1}) \sim p(x, y)$ and $x'_i, x'_{i+1} \sim p(x)$
    Update $w \leftarrow w - \eta_t \partial_w \bar{l}(x_i, x_{i+1}, y_i, y_{i+1}, x'_i, x'_{i+1}, f)$ where $f(x) = \langle \phi(x), w \rangle$
    Project $w$ onto $A$ via $w \leftarrow \operatorname{argmin}_{\bar{w} \in A} \|w - \bar{w}\|$.
**end for**

---

*Remark:* The streaming formulation does not impose any in-principle limitation regarding matching sample sizes. The only difference is that in the unmatched case we want to give samples from both distributions different weights (1/m and 1/m' respectively), e.g. by modifying the sampling procedure (see Table 3, Section 5).

**DC Programming** Alternatively, the Concave Convex Procedure, best known as DC programming in optimization [2], can be used to find an approximate solution of the problem in (5) by solving a succession of convex programs. DC programming has been used extensively in almost any other transductive algorithms to deal with non-convexity of the objective function. It works as follows: for a given function $F(x)$ that can be written as a difference of two convex functions $G$ and $H$ via $F(x) = G(x) - H(x)$, the below inequality

$$F(x) \leq \bar{F}(x, x_0) := G(x) - H(x_0) - \langle x - x_0, \partial_x H(x_0) \rangle \tag{16}$$

holds for all $x_0$ with equality for $x = x_0$, due to the convexity of $H(x)$. This implies an iterative algorithm for finding a local minimum of $F$ by minimizing the upper bound $\bar{F}(x, x_0)$ and subsequently updating $x_0 \leftarrow \operatorname{argmin}_x F(x, x_0)$ to the minimizer of the upper bound.

In order to minimize an additively decomposable objective function as in our transductive estimation, we could use stochastic gradient descent on the convex upper bound. Note that here the convex upper bound is given by a sum over the convex upper bounds for all terms. This strategy, however, is deficient in a significant aspect: the convex upper bounds on each of the loss terms become increasingly loose as we move $f$ away from the current point of approximation. It would be considerably better if we updated the upper bound after every stochastic gradient descent step. This variant, however, is *identical* to stochastic gradient descent on the original objective function due to the following:

$$\partial_x F(x)|_{x=x_0} = \partial_x \bar{F}(x, x_0)|_{x=x_0} = \partial_x G(x)|_{x=x_0} - \partial_x H(x)|_{x=x_0} \text{ for all } x_0. \tag{17}$$

In other words, in order to compute the gradient of the upper bound we need not compute the upper bound itself. Instead we may use the nonconvex objective directly, hence we did not pursue DC programming approach and Algorithm 1 applies.

## 5 Experiments

To demonstrate the applicability of our approach, we apply transduction to binary and multiclass classification both on toy datasets from the UCI repository [16] and the LibSVM site [17], plus

a larger scale multi-category classification dataset with $3.2 \cdot 10^6$ observations. We also perform experiments on a structured estimation problem, i.e. Japanese named entity recognition task and CoNLL-2000 base NP chunking task.

**Algorithms**   Since we are not aware of other transductive algorithms which can be applied easily to *all* the problems we consider, we choose problem-specific transduction algorithms as competitors. Multi Switch Transductive SVM (**MultiSwitch**) is used for *binary* classification [14]. This method is a variant of transductive SVM algorithm [8] tailored for linear semi-supervised binary classification on large and sparse datasets and involves switching of more than a single pair of labels at a time. For *multiclass* categorization we pick a Gaussian processes based transductive algorithm with distribution matching term (**GPDistMatch**) [5].

We use stochastic gradient descent for optimization in both inductive and transductive settings for binary and multiclass losses. More specifically, for transduction we use the Gaussian RBF kernel to compare distributions in (14). Note that, in the multiclass case, the additional distribution matching term measures the distance between multivariate functions.

**Small Scale Experiments**   We used the following datasets: binary (breastcancer, derm, optdigits, wdbc, ionosphere, iris, specft, pageblock, tae, heart, splice, adult, australian, bupa, cmc, german, pima, tic, yeast, sonar, cleveland, svmguide3 and musk) from the UCI repository and multiclass (usps, satimage, segment, svmguide2, vehicle). The data was preprocessed to have zero mean and unit variance.

Since we anticipate the relevant length scale in the margin distribution to be in the order of 1 (after all, we use a loss function, i.e. a hinge loss, which uses a margin of 1) we pick a Gaussian RBF kernel width of 0.2 for binary classification. Moreover, to take scaling in the number of classes into account we choose a kernel width of $0.1\sqrt{c}$ for multicategory classification. Here $c$ denotes the number of classes. We could indeed vary this width but we note in our experiments that the proposed method is not sensitive to this kernel width.

We split data equally into training and test sets, performing model selection on the training set and assessing performance on the test set. In these small scale experiments, we tune hyperparameters via 5-fold cross validation on the entire training set. The whole procedure was then repeated 5 times to obtain confidence bounds. More specifically, in the model selection stage, for transduction we adjust the regularization $\lambda$ and the transductive weight term $\gamma$ (obviously, for inductive inference we only need to adjust $\lambda$). For MultiSwitch Transduction the positive class fraction of unlabeled data was estimated using the training set [14]. Likewise, the two associated regularization parameters were tuned on the training set. For GP transduction both the regularization and divergence parameters were adjusted.

**Results**   The experimental results are summarized in Figure 2 for a binary setting and in Table 1 for a multiclass problem. In 23 binary datasets, transduction outperforms the inductive setup in 20 of them. Arguably, our proposed transductive method performs on a par with state-of-the-art transductive approach for each learning problem. In the binary estimation, out of 23 datasets, our method performs significantly worse than MultiSwitch transduction algorithm in 4 datasets (adult, bupa, pima, and svmguide3) and significantly better on 2 datasets (ionosphere and pageblock), using a one-sided paired t-test with 95% confidence. Overall, both algorithms are very comparable. The advantage of our approach is that it is 'plug and play', i.e. for different problems we only need to use the appropriate supervised loss function. The distribution matching penalty itself remains unchanged. Further, by casting the transductive solution as an online optimization method, our approach scales well.

**Larger Scale Experiments**   Since one of the key points of our approach is that it *can* be applied to large problems, we performed transduction on the DMOZ ontology [20] of topics. We selected the top 2 levels of the topic tree (575) and removed all but the 100 most frequent ones, since a large number of topics occurs only very rarely. This left us with 89.2% of the initial webpages. As feature vectors we used the standard bag of words representation of the web page descriptions with TF-IDF weighting. The dictionary size (and therefore the dimensionality of our features) is

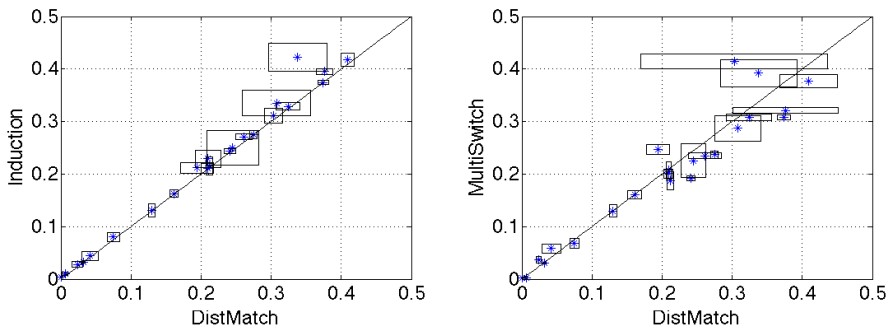

Figure 2: Error rate on 23 *binary* estimation problems. Left panel, DistMatch against Induction; Right panel, DistMatch against MultiSwitch. `DistMatch:` distribution matching (ours) and `MultiSwitch:` Multi switch transductive SVM, [14]. Height of the box encodes standard error of DistMatch and width of the box encodes standard error of Induction / MultiSwitch.

Table 1: Error rate $\pm$ standard deviation on a *multi-category* estimation problem. `DistMatch:` distribution matching (ours) and `GPDistMatch:` Gaussian Process transduction, [5].

| dataset | $m$ | classes | Induction | DistMatch | GPDistMatch |
|---|---|---|---|---|---|
| usps | 730 | 10 | 0.143±0.021 | 0.125±0.019 | 0.140±0.034 |
| satimage | 620 | 6 | 0.190±0.052 | 0.186±0.037 | 0.212±0.034 |
| segment | 693 | 7 | 0.279±0.090 | 0.206±0.047 | 0.181±0.020 |
| svmguide2 | 391 | 3 | 0.280±0.028 | 0.256±0.020 | 0.231±0.018 |
| vehicle | 423 | 4 | 0.385±0.070 | 0.333±0.048 | 0.336±0.060 |

Table 2: Error rate on the *DMOZ ontology* for increasing training / test set sizes.

| training / test set size | 50,000 | 100,000 | 200,000 | 400,000 | 800,000 | 1,600,000 |
|---|---|---|---|---|---|---|
| induction | 0.365 | 0.362 | 0.337 | 0.299 | 0.300 | 0.268 |
| transduction | 0.344 | 0.326 | 0.330 | 0.288 | 0.263 | 0.250 |

Table 3: Error rate on the *DMOZ ontology* for fixed training set size of 100,000 samples.

| test set size | 100,000 | 200,000 | 400,000 | 800,000 | 1,600,000 |
|---|---|---|---|---|---|
| induction | 0.358 | 0.358 | 0.357 | 0.357 | 0.357 |
| transduction | 0.326 | 0.316 | 0.306 | 0.322 | 0.329 |

Table 4: Accuracy, precision, recall and $F_{\beta=1}$ score on the *Japanese named entity* task.

| | Accuracy | Precision | Recall | F1 Score |
|---|---|---|---|---|
| induction | 96.82 | 84.15 | 72.49 | 77.89 |
| transduction | 97.13 | 84.46 | 75.30 | 79.62 |

Table 5: Accuracy, precision, recall and $F_{\beta=1}$ score on the *CoNLL-2000 base NP chunking* task.

| | Accuracy | Precision | Recall | F1 Score |
|---|---|---|---|---|
| induction | 95.72 | 90.99 | 90.72 | 90.85 |
| transduction | 96.05 | 91.73 | 91.97 | 91.85 |

1,319,489. For these larger scale experiments, we use a dataset of up to $3.2 \cdot 10^6$ observations. To our knowledge, our proposed transduction method is the only one that scales very well due to the stochastic approximation.

For each experiment, we split data into training and test sets. Model selection is perform on the training set by putting aside part of the training data as a validation set which is then used exclusively for tuning the hyperparameters. In large scale transduction two issues matter: firstly, the algorithm needs to be scalable with respect to the training set size. Secondly, we need to be able to scale the algorithm with respect to the test set. Both results can be seen in Tables 2 and 3. Note that Table 2 uses an equal split between training and test sets, while Table 3 uses an unequal split where the test

set has many more observations. We see that the algorithm improves with increasing data size, both for training and test sets. In the latter case, only up to some point: for the larger test sets (800,000 and 1,600,000) it decreases (although still stays better than inductive's). We suspect that a location-dependent transduction score would be useful in this context – i.e. instead of only minimizing the discrepancy between decision function values on training and test set $D(f(X), f(X'))$ we could also introduce local features $D((X, f(X)), (X', f(X')))$.

**Japanese Named Entity Recognition Experiments** A key advantage of our transduction algorithm is it *can* be applied to structured estimation without modification. We used the Japanese named-entity recognition dataset provided with the CRF++ toolkit [18]. The data contains 716 Japanese sentences with 17 annotated named entities. The task is to detect and classify proper nouns and numerical information in a document into categories such as names of persons, organizations, locations, times and quantities. Conditional random fields (CRFs) [9] are considered to be the state-of-the-art framework for this sequential labeling problem [11].

As the basis of our implementation we used Leon Bottou's CRF code [19]. We use simple 1D chain CRFs with first order Markov dependency between name tags. That is, we have clique potentials joining adjacent labels $(y_i, y_{i+1})$, but which are independent of the text itself, and clique potentials joining words and labels $(x_i, y_i)$. Since the former do not depend on the test data there is no need to enforce distribution matching. For the latter, though, we want to enforce that clique potentials are distributed in the same way between training and test set. The stationarity assumption in the potentials implies that this needs to hold uniformly over all such cliques.

Since the number of tokens per sentence is variable, i.e. the chain length itself is a random variable, we perform distribution matching on a per-token basis — we oversample each token 10 times in our experiments. This strikes a balance between statistical accuracy and computational efficiency. The additional distribution matching term is then measuring the distance between these over-sampled clique potentials. As before, we split data equally into training and test sets and put aside part of the training data as a validation set which is used exclusively for tuning the hyperparameters. We relied on the feature template provided in CRF++ for this task. We report results in Table 4, that is precision (fraction of name tags which match the reference tags), recall (fraction of reference tags returned), and their harmonic mean, $F_{\beta=1}$ are reported. Transduction outperforms induction in all metrics.

**CoNLL-2000 Base NP Chunking Experiments** Our second structured estimation experiment is the CoNLL-2000 base NP chunking dataset [13] as provided in the CRF++ toolkit. The task is to divide text into syntactically correlated parts. The dataset has 900 sentences and the goal is to label each word with a label indicating whether the word is outside a chunk, starts a chunk, or continues a chunk.

Similarly to Japanese named entity recognition task, 1D chain CRFs with only first order Markov dependency between chunk tags are modeled. We considered binary-valued features which depend on the words, part-of-speech tags, and labels in the neighborhood of a given word as encoded in the CRF++ feature template. The same experimental setup as in named entity experiments is used. The results in terms of accuracy, precision, recall and F1 score are summarized in Table 5. Again, transduction outperforms the inductive setup.

## 6 Summary and Discussion

We proposed a transductive estimation algorithm which is a) simple, b) general c) scalable and d) works well when compared to the state of the art algorithms applied to each *specific* problem. Not only is it useful for classical binary and multiclass categorization problems but it also applies to ontologies and structured estimation problems. It is not surprising that it performs very comparably to existing algorithms, since they can, in many cases, be seen as special instances of the general purpose distribution matching setting.

Extensions of distribution matching beyond simply modeling $f(X)$ and instead, modeling $(X, f(X))$, that is, the introduction of local features, obtaining good theoretical bounds on the shrinkage of the function class via the distribution matching constraint, and applications to other function classes (e.g. balancing decision trees) are subject of future research.

# References

[1] O. Chapelle, B. Schölkopf, and A. Zien, editors. *Semi-Supervised Learning*. MIT Press, Cambridge, MA, 2006.

[2] T. Pham Dinh and L. Hoai An. A D.C. optimization algorithm for solving the trust-region subproblem. *SIAM Journal on Optimization*, 8(2):476–505, 1988.

[3] G. Druck, G.S. Mann, and A. McCallum. Learning from labeled features using generalized expectation criteria. In S.-H. Myaeng, D.W. Oard, F. Sebastiani, T.-S. Chua, and M.-K. Leong, editors, *SIGIR*, pages 595–602. ACM, 2008.

[4] A. Gammerman, Volodya Vovk, and Vladimir Vapnik. Learning by transduction. In *Proceedings of Uncertainty in AI*, pages 148–155, Madison, Wisconsin, 1998.

[5] T. Gärtner, Q.V. Le, S. Burton, A. J. Smola, and S. V. N. Vishwanathan. Large-scale multiclass transduction. In Y. Weiss, B. Schölkopf, and J. Platt, editors, *Advances in Neural Information Processing Systems 18*, pages 411–418, Cambride, MA, 2006. MIT Press.

[6] J. Graça, K. Ganchev, and B. Taskar. Expectation maximization and posterior constraints. In J. C. Platt, D. Koller, Y. Singer, and S. T. Roweis, editors, *NIPS*. MIT Press, 2007.

[7] A. Gretton, K. Borgwardt, M. Rasch, B. Schölkopf, and A. Smola. A kernel method for the two sample problem. Technical Report 157, MPI for Biological Cybernetics, 2008.

[8] T. Joachims. Transductive inference for text classification using support vector machines. In I. Bratko and S. Dzeroski, editors, *Proc. Intl. Conf. Machine Learning*, pages 200–209, San Francisco, 1999. Morgan Kaufmann Publishers.

[9] J. D. Lafferty, A. McCallum, and F. Pereira. Conditional random fields: Probabilistic modeling for segmenting and labeling sequence data. In *Proc. Intl. Conf. Machine Learning*, volume 18, pages 282–289, San Francisco, CA, 2001. Morgan Kaufmann.

[10] Q.V. Le, A.J. Smola, T. Gärtner, and Y. Altun. Transductive gaussian process regression with automatic model selection. In J. Fürnkranz, T. Scheffer, and M. Spiliopoulou, editors, *European Conference of Machine Learning*, volume 4212 of *LNAI*. 306-317, 2006.

[11] A. McCallum and W. Li. Early results for named entity recognition with conditional random fields, feature induction and web enhanced lexicons. In *CoNLL*, 2003.

[12] Y. Nesterov and J.-P. Vial. Confidence level solutions for stochastic programming. Technical Report 2000/13, Université Catholique de Louvain - Center for Operations Research and Economics, 2000.

[13] E.F. Tjong Kim Sang and S. Buchholz. Introduction to the CoNLL-2000 shared task: Chunking. In *Proc. Conf. Computational Natural Language Learning*, pages 127–132, Lisbon, Portugal, 2000.

[14] V. Sindhwani and S.S. Keerthi. Large scale semi-supervised linear SVMs. In *SIGIR '06: Proceedings of the 29th annual international ACM SIGIR conference on Research and development in information retrieval*, pages 477–484, New York, NY, USA, 2006. ACM Press.

[15] A. Zien, U. Brefeld, and T. Scheffer. Transductive support vector machines for structured variables. In *ICML*, pages 1183–1190, 2007.

[16] UCI repository, `http://archive.ics.uci.edu/ml/`

[17] LibSVM, `http://www.csie.ntu.edu.tw/~cjlin/libsvmtools/`

[18] CRF++, `http://chasen.org/~taku/software/CRF++`

[19] Stochastic Gradient Descent code, `http://leon.bottou.org/projects/sgd`

[20] DMOZ ontology, `http://www.dmoz.org`

